# Parallel Sampling of DP Mixture Models using Sub-Clusters Splits

**Jason Chang**[*]
CSAIL, MIT
jchang7@csail.mit.edu

**John W. Fisher III**[*]
CSAIL, MIT
fisher@csail.mit.edu

## Abstract

We present an MCMC sampler for Dirichlet process mixture models that can be parallelized to achieve significant computational gains. We combine a non-ergodic, restricted Gibbs iteration with split/merge proposals in a manner that produces an ergodic Markov chain. Each cluster is augmented with two sub-clusters to construct likely split moves. Unlike some previous parallel samplers, the proposed sampler enforces the correct stationary distribution of the Markov chain without the need for finite approximations. Empirical results illustrate that the new sampler exhibits better convergence properties than current methods.

## 1 Introduction

Dirichlet process mixture models (DPMMs) are widely used in the machine learning community (e.g. [28, 32]). Among other things, the elegant theory behind DPMMs has extended finite mixture models to include automatic model selection in clustering problems. One popular method for posterior inference in DPMMs is to draw samples of latent variables using a Markov chain Monte Carlo (MCMC) scheme. Extensions to the DPMM such as the Hierarchical Dirichlet processes [29] and the dependent Dirichlet process [18] also typically employ sampling-based inference.

Posterior sampling in complex models such as DPMMs is often difficult because samplers that propose local changes exhibit poor convergence. Split and merge moves, first considered in DPMMs by [13], attempt to address these convergence issues. Alternatively, approximate inference methods such as the variational algorithms of [3] and [15] can be used. While variational algorithms do not have the limiting guarantees of MCMC methods and may also suffer from similar convergence issues, they are appealing for use in large datasets as they lend themselves to parallelization. Here, we develop a sampler for DPMMs that: (1) preserves limiting guarantees; (2) proposes splits and merges to improve convergence; (3) can be parallelized to accommodate large datasets; and (4) is applicable to a wide variety of DPMMs (conjugate and non-conjugate). To our knowledge, no current sampling algorithms satisfy all of these properties simultaneously.

While we focus on DP mixture models here, similar methods can be extended for mixture models with other priors (finite Dirichlet distributions, Pitman-Yor Processes, etc.).

## 2 Related Work

Owing to the wealth of literature on DPMM samplers, we focus on the most relevant work in our overview. Other sampling algorithms (e.g. [17]) and inference methods (e.g. [3]) are not discussed. The majority of DPMM samplers fit into one of two categories: collapsed-weight samplers that

---

[*]Jason Chang was partially supported by the Office of Naval Research Multidisciplinary Research Initiative (MURI) program, award N000141110688. John Fisher was partially supported by the Defense Advanced Research Projects Agency, award FA8650-11-1-7154.

Table 1: Capabilities of MCMC Sampling Algorithms

| | CW | [11, 12] | [7, 24] | [5, 9, 13] | [14] | [19, 31] | Proposed Method |
|---|---|---|---|---|---|---|---|
| Exact Model | ✓ | · | ✓ | ✓ | ✓ | ✓ | ✓ |
| Splits & Merges | · | · | · | ✓ | ✓ | · | ✓ |
| Intra-cluster Parallelizable | · | · | · | · | · | ✓ | ✓ |
| Inter-cluster Parallelizable | · | ✓ | ✓ | · | · | · | ✓ |
| Non-conjugate Priors | ✓ | ✓ | ✓ | · | ✓ | · | ✓ |

marginalize over the mixture weights or instantiated-weight samplers that explicitly represent them. Capabilities of current algorithms, which we now discuss, are summarized in Table 1.

Collapsed-weight (CW) samplers using both conjugate (e.g. [4, 6, 20, 22, 30]) and non-conjugate (e.g. [21, 23]) priors sample the cluster labels iteratively one data point at a time. When a conjugate prior is used, one can also marginalize out cluster parameters. However, as noted by multiple authors (e.g. [5, 13, 17]), these methods often exhibit slow convergence. Additionally, due to the particular marginalization schemes, these samplers cannot be parallelized.

Instantiated-weight (IW) samplers explicitly represent cluster weights, typically using a finite approximation to the DP (e.g. [11, 12]). Recently, [7] and [24] have eliminated the need for this approximation; however, IW samplers still suffer from convergence issues. If cluster parameters are marginalized, it can be very unlikely for a single point to start a new cluster. When cluster parameters are instantiated, samples of parameters from the prior are often a poor fit to the data. However, IW samplers are often useful because they can be parallelized across each data point conditioned on the weights and parameters. We refer to this type of algorithm as "inter-cluster parallelizable", since the cluster label for each point within a cluster can be sampled in parallel.

The recent works of [19] and [31] present an alternative parallelization scheme for CW samplers. They observe that multiple clusters can be grouped into "super-clusters" and that each super-cluster can be sampled independently. We refer to this type of implementation as "intra-cluster parallelizable", since points in different super-clusters can be sampled in parallel, but points within a cluster cannot. This distinction is important as many problems of interest contain far more data points than clusters, and the greatest computational gain may come from inter-cluster parallelizable algorithms. Due to their particular construction, current algorithms group super-clusters solely based on the size of each super-cluster. In the sequel, we show empirically that this can lead to slow convergence and demonstrate how data-based super-clusters improve upon these methods.

Recent CW samplers consider larger moves to address convergence issues. Green and Richardson [9] present a reversible jump MCMC sampler that proposes splitting and merging components. While a general framework is presented, proposals are model-dependent and generic choices are not specified. Proposed splits are unlikely to fit the posterior since auxiliary variables governing the split cluster parameters and weights are proposed independent of the data. Jain and Neal [13, 14] construct a split by running multiple restricted Gibbs scans for a single cluster in conjugate and non-conjugate models. While each restricted scan improves the constructed split, it also increases the amount of computation needed. As such, it is not easy to determine how many restricted scans are needed. Dahl [5] proposes a split scheme for conjugate models by reassigning labels of a cluster sequentially. All current split samplers construct a proposed move to be used in a Metropolis-Hastings framework. If the split is rejected, considerable computation is wasted, and all information contained in learning the split is forgotten. In contrast, the proposed method of fitting sub-clusters iteratively learns likely split proposals with the auxiliary variables. Additionally, we show that split proposals can be computed in parallel, allowing for very efficient implementations.

## 3 Dirichlet Process Mixture Model Samplers

In this section we give a brief overview of DPMMs. For a more in-depth understanding, we refer the reader to [27]. A graphical model for the DPMM is shown in Figure 1a, where $i$ indexes a particular data point, $x$ is the vector of observed data, $z$ is the vector of cluster indices, $\pi$ is the infinite vector of mixture weights, $\alpha$ is the concentration parameter for the DP, $\theta$ is the vector of the cluster parameters, and $\lambda$ is the hyperparameter for the corresponding DP base measure.

### 3.1 Instantiated-Weight Samplers using Approximations to the Dirichlet Process

The constructive proof of the Dirichlet processes [26] shows that a DP can be sampled by iteratively scaling an infinite sequence of Beta random variables. Therefore, posterior MCMC inference in a DPMM could, in theory, alternate between the following samplers

$$(\pi_1, \ldots, \pi_\infty) \sim p(\pi|z, \alpha), \tag{1}$$

$$\theta_k \overset{\propto}{\sim} f_x(x_{\{k\}}; \theta_k) f_\theta(\theta_k; \lambda), \qquad \forall k \in \{1, \ldots, \infty\}, \tag{2}$$

$$z_i \overset{\propto}{\sim} \sum_{k=1}^\infty \pi_k f_x(x_i; \theta_k) \mathbb{I}[z_i = k], \qquad \forall i \in \{1, \ldots, N\}, \tag{3}$$

where $\overset{\propto}{\sim}$ samples from a distribution proportional to the right side, $x_{\{k\}}$ denotes the (possibly empty) set of data labeled $k$, and $f_\circ(\cdot)$ denotes a particular form of the probability density function of $\circ$. We use $f_x(x_{\{k\}}; \theta_k)$ to denote the product of likelihoods for all data points in cluster $k$. When conjugate priors are used, the posterior distribution for cluster parameters is in the same family as the prior:

$$p(\theta_k|x, z, \lambda) \propto f_\theta(\theta_k; \lambda) f_x(x_{\{k\}}; \theta_k) \propto f_\theta(\theta_k; \lambda_k^*), \tag{4}$$

where $\lambda_k^*$ denotes the posterior hyperparameters for cluster $k$. Unfortunately, the infinite length sequences of $\pi$ and $\theta$ clearly make this procedure impossible.

As an approximation, authors have considered the truncated stick-breaking representation [11] and the finite symmetric Dirichlet distribution [12]. These approximations become more accurate when the truncation is much larger than the true number of components. However, knowledge of the true number of clusters is often unknown. When cluster parameters are explicitly sampled, these algorithms may additionally suffer from slow convergence issues. In particular, a broad prior will often result in a very small probability of creating new clusters since the probability of generating a parameter from the prior to fit a single data point is small.

### 3.2 Collapsed-Weight Samplers using the Chinese Restaurant Process

Alternatively, the weights can be marginalized to form a collapsed-weight sampler. By exchangeability, a label can be drawn using the Chinese Restaurant Process (CRP) [25], which assigns a new customer (i.e. data point) to a particular table (i.e. cluster) with the following predictive distribution

$$p(z_i|x, z_{\setminus i}; \alpha) \propto \left[ \sum_k N_{k \setminus i} f_x(x_i; \lambda_{k \setminus i}^*) \mathbb{I}[z = k] \right] + \alpha f_x(x_i; \lambda) \mathbb{I}[z = \hat{k}], \tag{5}$$

where $\setminus i$ denotes all indices excluding $i$, $N_{k \setminus i}$ are the number of elements in $z_{\setminus i}$ with label $k$, $\hat{k}$ is a new cluster label, and $f_x(\circ; \lambda)$ denotes the distribution of $x$ when marginalizing over parameters. When a non-conjugate prior is used, a computationally expensive Metropolis-Hastings step (e.g. [21, 23]) must be used when sampling the label for each data point.

## 4 Exact Parallel Instantiated-Weight Samplers

We now present a novel alternative to the instantiated-weight samplers that does not require any finite model approximations. The *detailed balance* property underlies most MCMC sampling algorithms. In particular, if one desires to sample from a target distribution, $\pi(z)$, satisfying detailed balance for an *ergodic* Markov chain guarantees that simulations of the chain will uniquely converge to the target distribution of interest. We now consider the atypical case of simulating from a *non-ergodic* chain with a transition distribution that satisfies detailed balance.

**Definition 4.1** (Detailed Balance). *Let $\pi(z)$ denote the target distribution. If a Markov chain is constructed with a transition distribution $q(\hat{z}|z)$ that satisfies $\pi(z)q(\hat{z}|z) = \pi(\hat{z})q(z|\hat{z})$, then the chain is said to satisfy the detailed balance condition and $\pi(z)$ is guaranteed to be a stationary distribution of the chain.*

We define a *restricted* sampler as one that satisfies detailed balance (e.g. using the Hastings ratio [10]) but does not result in an ergodic chain. We note that without ergodicity, detailed balance does not imply uniqueness in, or convergence to the stationary distribution. One key observation of this work is that multiple restricted samplers can be combined to form an ergodic chain. In particular,

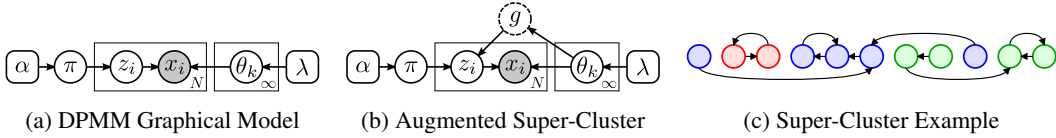

|  (a) DPMM Graphical Model  |  (b) Augmented Super-Cluster  |  (c) Super-Cluster Example  |

Figure 1: (a)-(b) Graphical models for the DPMM and augmented super-cluster space. Auxiliary variables are dotted. (c) An illustration of the super-cluster grouping. Nodes represent clusters, arrows point to neighbors, and colors represent the implied super-clusters.

we consider a sampler that is restricted to only sample labels belonging to non-empty clusters. Such a sampler is not ergodic because it cannot create new clusters. However, when mixed with a sampler that proposes splits, the resulting chain is ergodic and yields a valid sampler. We now consider a restricted *Gibbs* sampler. The coupled split sampler is discussed in Section 5.

## 4.1 Restricted DPMM Gibbs Sampler with Super-Clusters

A property stemming from the definition of Dirichlet processes is that the measure for every finite partitioning of the measurable space is distributed according to a Dirichlet distribution [8]. While the DP places an infinite length prior on the labels, any realization of $z$ will belong to a finite number of clusters. Supposing $z_i \in \{1, \cdots, K\}$, $\forall i$, we show in the supplement that the posterior distribution of mixture weights, $\pi$, conditioned on the cluster labels can be expressed as

$$(\pi_1, \cdots, \pi_K, \tilde{\pi}_{K+1}) \sim \mathrm{Dir}\left(N_1, \cdots, N_K, \alpha\right), \tag{6}$$

where $N_k = \sum_i \mathbb{I}[z_i = k]$ is the number of points in cluster $k$, and $\tilde{\pi}_{K+1} = \sum_{k=K+1}^{\infty} \pi_k$ is the sum of all empty mixture weights. This relationship has previously been noted in the literature (c.f. [29]). In conjunction with Definition 4.1, this leads to the following iterated restricted Gibbs sampler:

$$(\pi_1, \ldots, \pi_K, \tilde{\pi}_{K+1}) \sim \mathrm{Dir}(N_1, \ldots, N_K, \alpha), \tag{7}$$

$$\theta_k \overset{\propto}{\sim} f_x(x_{\{k\}}; \theta_k) f_\theta(\theta_k; \lambda), \qquad \forall k \in \{1, \ldots, K\}, \tag{8}$$

$$z_i \overset{\propto}{\sim} \sum_{k=1}^{K} \pi_k f_x(x_i; \theta_k) \mathbb{I}[z_i = k], \quad \forall i \in \{1, \ldots, N\}. \tag{9}$$

We note that each of these steps can be parallelized and, because the mixture parameters are explicitly represented, this procedure works for conjugate and non-conjugate priors. When non-conjugate priors are used, any proposal that leaves the stationary distribution invariant can be used (c.f. [23]).

Similar to previous super-cluster methods, we can also restrict each cluster to only consider moving to a subset of other clusters. The super-clusters of [19] and [31] are formed using a size-biased sampler. This can lead to slower convergence since clusters with similar data may not be in the same super-cluster. By observing that *any* similarly restricted Gibbs sampler satisfies detailed balance, any randomized algorithm that assigns finite probability to any super-cluster grouping can be used. As shown in Figure 1b, we augment the sample space with super-cluster groups, $g$, that group similar clusters together. Conditioned on $g$, Equation 9 is altered to only consider labels within the super-cluster that the data point currently belongs to. The super-cluster sampling procedure is described in Algorithm 1. Here, $D$ denotes an arbitrary distance measure between probability distributions. In our experiments, we use the symmetric version of KL-divergence (J-divergence). When the J-divergence is difficult to calculate, any distance measure can be substituted. For example, in the case of multinomial distributions, we use the J-divergence for the categorical distribution as a proxy. An illustration of the implied super-cluster grouping from the algorithm is shown in Figure 1c and a visualization of an actual super-cluster grouping is shown in Figure 2. Notice that the super-cluster groupings using [19] are essentially random while our super-clusters are grouped by similar data.

---

**Algorithm 1** Sampling Super-clusters with Similar Cluster

---

1. Form the adjacency matrix, $A$, where $A_{k,m} = \exp[-D(f_x(\circ; \theta_k), f_x(\circ; \theta_m))]$
2. For each cluster, $k$, sample a random neighbor $k'$, according to, $k' \overset{\propto}{\sim} \sum_m A_{k,m} \mathbb{I}[k' = m]$
3. Form the groups of super-clusters, $g$, by finding the separate connected graphs

---

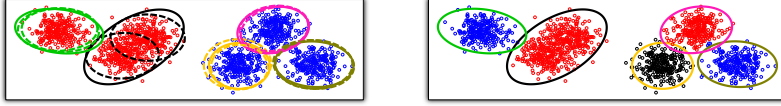

Figure 2: (left) A visualization of the algorithm. Each set of uniquely colored ellipses indicate one cluster. Solid ellipses indicate regular clusters and dotted ellipses indicate sub-cluster. Color of data points indicate super-cluster membership. (right) Inferred clusters and super-clusters from [19].

## 5 Parallel Split/Merge Moves via Sub-Clusters

The preceding section showed that an exact MCMC sampling algorithm can be constructed by alternating between a restricted Gibbs sampler and split moves. While any split proposal (e.g. [5, 13, 14]) can result in an ergodic chain, we now develop efficient split moves that are compatible with conjugate and non-conjugate priors and that can be parallelized. We will augment the space with auxiliary variables, noting that samples of the non-auxiliary variables can be obtained by drawing samples from the joint space and simply discarding any auxiliary values.

### 5.1 Augmenting the Space with Auxiliary Variables

Since the goal is to design a model that is tailored toward splitting clusters, we augment each regular cluster with two explicit sub-clusters (herein referred to as the "left" and "right" sub-clusters). Each data point is then attributed with a sub-cluster label, $\overline{z}_i \in \{\ell, r\}$, indicating whether it comes from the left or right sub-cluster. Additionally, each sub-cluster has an associated pair of weights, $\overline{\pi}_k = \{\overline{\pi}_{k,\ell}, \overline{\pi}_{k,r}\}$, and parameters, $\overline{\theta}_k = \{\overline{\theta}_{k,\ell}, \overline{\theta}_{k,r}\}$. These auxiliary variables are named in a similar fashion to their regular-cluster counterparts because of the similarities between sub-clusters and regular-clusters. One naïve choice for auxiliary parameter distributions is

$$p(\overline{\pi}_k) = \text{Dir}(\overline{\pi}_{k,\ell}, \overline{\pi}_{k,r}; \alpha/2, \alpha/2), \qquad p(\overline{\theta}_k) = f_\theta(\overline{\theta}_{k,\ell}; \lambda) f_\theta(\overline{\theta}_{k,r}; \lambda), \qquad (10)$$

$$p(\overline{z}|\overline{\pi}, \theta, x, z) = \prod_k \prod_{\{i; z_i = k\}} \frac{\overline{\pi}_{k,\overline{z}_i} f_x(x_i; \overline{\theta}_{k,\overline{z}_i})}{\overline{\pi}_{k,\ell} f_x(x_i; \overline{\theta}_{k,\ell}) + \overline{\pi}_{k,r} f_x(x_i; \overline{\theta}_{k,r})}. \qquad (11)$$

The corresponding graphical model is shown in Figure 3a. It would be advantageous if the form of the posterior for the auxiliary variables matched those of the regular-clusters in Equation 7-9. Unfortunately, because the normalization in Equation 11 depends on $\overline{\pi}$ and $\overline{\theta}$, this choice of auxiliary distributions does not result in the posterior distributions for $\overline{\pi}$ and $\overline{\theta}$ that one would expect. We note that this problem only arises in the auxiliary space where $x$ generates the auxiliary label $\overline{z}$ (in contrast to the regular space, where $z$ generates $x$). Additional details are provided in the supplement.

Consequently, we alter the distribution over sub-cluster parameters to be

$$p(\overline{\theta}_k|x, z, \pi) \propto f_\theta(\overline{\theta}_{k,\ell}; \lambda) f_\theta(\overline{\theta}_{k,r}; \lambda) \prod_{\{i; z_i = k\}} \left( \overline{\pi}_{k,\ell} f_x(x_i; \overline{\theta}_{k,\ell}) + \overline{\pi}_{k,r} f_x(x_i; \overline{\theta}_{k,r}) \right). \qquad (12)$$

It is easily verified that this choice results in the the following conditional posterior distributions

$$(\overline{\pi}_{k,\ell}, \overline{\pi}_{k,r}) \sim \text{Dir}(N_{k,\ell} + \alpha/2, N_{k,r} + \alpha/2), \qquad \forall k \in \{1, \dots, K\}, \qquad (13)$$

$$\overline{\theta}_{k,s} \overset{\propto}{\sim} f_x(x_{\{k,s\}}; \overline{\theta}_{k,s}) f_\theta(\overline{\theta}_{k,s}; \lambda), \qquad \forall k \in \{1, \dots, K\}, \forall s \in \{\ell, r\}, \qquad (14)$$

$$\overline{z}_i \overset{\propto}{\sim} \sum_{s \in \{\ell, r\}} \overline{\pi}_{z_i, s} f_x(x_i; \overline{\theta}_{z_i, s}) \mathbb{1}[\overline{z}_i = s], \qquad \forall i \in \{1, \dots, N\}, \qquad (15)$$

which essentially match the distributions for regular-cluster parameters in Equation 7-9. We note that the joint distribution over the augmented space cannot be expressed analytically as a result of only specifying Equation 12 up to a proportionality constant that depends on $\overline{\pi}$, $x$, and $z$. The corresponding graphical model is shown in Figure 3b.

### 5.2 Restricted Gibbs Sampling in Augmented Space

Restricted sampling in the augmented space can be performed in a similar fashion as before. One can draw a sample from the space of $K$ regular clusters by sampling all the regular- and sub-cluster parameters conditioned on labels and data from Equations 7, 8, 13, and 14. Conditioned on these parameters, one can sample a regular-cluster label followed by a sub-cluster label for each data point from Equations 9 and 15. All of these steps can be computed in parallel.

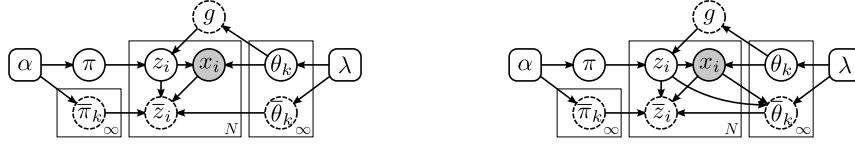

(a) Unmatched Augmented Sub-Cluster Model      (b) Matched Augmented Sub-Cluster Model

Figure 3: Graphical models for the augmented DPMMs. Auxiliary variables are dotted.

## 5.3 Metropolis-Hastings Sub-Cluster Split Moves

A pair of inferred sub-clusters contains a likely split of the corresponding regular-cluster. We exploit these auxiliary variables to propose likely splits. Similar to previous methods, we use a Metropolis-Hastings (MH) MCMC [10] method for proposed splits. A new set of random variables, $\{\hat{\pi}, \hat{\theta}, \hat{z}, \hat{\overline{\pi}}, \hat{\overline{\theta}}, \hat{\overline{z}}\}$ are proposed via some proposal distribution, $q$, and accepted with probability

$$\min\left[1, \frac{p(\hat{\pi},\hat{z},\hat{\theta},x)p(\hat{\overline{\pi}},\hat{\overline{\theta}},\hat{\overline{z}}|x,\hat{z})}{p(\pi,z,\theta,x)p(\overline{\pi},\overline{\theta},\overline{z}|x,z)} \cdot \frac{q(\pi,z,\theta,\overline{\pi},\overline{\theta},\overline{z}|\hat{\pi},\hat{z},\hat{\theta},\hat{\overline{\pi}},\hat{\overline{\theta}},\hat{\overline{z}})}{q(\hat{\pi},\hat{z},\hat{\theta},\hat{\overline{\pi}},\hat{\overline{\theta}},\hat{\overline{z}}|\pi,z,\theta,\overline{\pi},\overline{\theta},\overline{z})}\right] = \min[1, H], \qquad (16)$$

where $H$ is the "Hastings ratio". Because of the required reverse proposal in the Hastings ratio, we must propose both merges and splits. Unfortunately, because the joint likelihood for the augmented space cannot be analytically expressed, the Hastings ratio for an arbitrary proposal distribution cannot be computed. A very specific proposal distribution, which we now discuss, does result in a tractable Hastings ratio. A split or merge move, denoted by $Q$, is first selected at random. In our examples, all possible splits and merges are considered since the number of clusters is much smaller than the number of data points. When this is not the case, any randomized proposal can be used. Conditioned on $Q = Q_{\text{split-}c}$, which splits cluster $c$ into $m$ and $n$, or $Q = Q_{\text{merge-}mn}$, which merges clusters $m$ and $n$ into $c$, a new set of variables are sampled with the following

$$Q = Q_{\text{split-}c} \qquad\qquad\qquad\qquad Q = Q_{\text{merge-}mn}$$
$$(\hat{z}_{\{m\}}, \hat{z}_{\{n\}}) = \text{split-}c(z, \overline{z}) \qquad\qquad \hat{z}_{\{c\}} = \text{merge-}mn(z) \qquad (17)$$
$$(\hat{\pi}_m, \hat{\pi}_n) = \pi_c \cdot (u_m, u_n), \quad (u_m, u_n) \sim \text{Dir}(\hat{N}_m, \hat{N}_n) \qquad \hat{\pi}_c = \hat{\pi}_m + \hat{\pi}_n \qquad (18)$$
$$(\hat{\theta}_m, \hat{\theta}_n) \sim q(\hat{\theta}_m, \hat{\theta}_n | x, \hat{z}, \hat{\overline{z}}) \qquad\qquad \hat{\theta}_c \sim q(\hat{\theta}_c | x, \hat{z}, \hat{\overline{z}}) \qquad (19)$$
$$\hat{\overline{v}}_m, \hat{\overline{v}}_n \sim p(\hat{\overline{v}}_m, \hat{\overline{v}}_n | x, \hat{z}) \qquad\qquad \hat{\overline{v}}_c \sim p(\hat{\overline{v}}_c | x, \hat{z}) \qquad (20)$$

Here, $\overline{v}_k = \{\overline{\pi}_k, \overline{\theta}_k, \overline{z}_{\{k\}}\}$ denotes the set of auxiliary variables for cluster $k$, the function split-$c(\circ)$ splits the labels of cluster $c$ based on the sub-cluster labels, and merge-$mn(\circ)$ merges the labels of clusters $m$ and $n$. The proposal of cluster parameters is written in a general form so that users can specify their own proposal for non-conjugate priors. All other cluster parameters remain the same. Sampling auxiliary variables from Equation 20 will be discussed shortly. Assuming that this can be performed, we show in the supplement that the resulting Hastings ratio for a split is

$$H_{\text{split-}c} = \frac{\alpha q(\theta_c|x,z,\hat{z})}{\Gamma(N_k)f_\theta(\theta_c;\lambda)f_x(x_{\{c\}};\theta_c)} \prod_{k\in\{m,n\}} \frac{\Gamma(\hat{N}_k)f_\theta(\hat{\theta}_k;\lambda)f_x(x_{\{k\}};\hat{\theta}_k)}{q(\hat{\theta}_k|x,z,\hat{z})} = \frac{\alpha \prod_{k\in\{m,n\}}\Gamma(\hat{N}_k)f_x(x_{\{k\}};\lambda)}{\Gamma(N_c)f_x(x_{\{c\}};\lambda)}. \quad (21)$$

The first expression can be used for non-conjugate models, and the second expression can be used in conjugate models where new cluster parameters are sampled directly from the posterior distribution. We note that these expressions do not have any residual normalization terms and can be computed exactly, even though the joint distribution of the augmented space can not be expressed analytically.

Unfortunately, the Hastings ratio for a merge move is slightly more complicated. We discuss these complications following the explanation of sampling the auxiliary variables in the next section.

## 5.4 Deferred Metropolis-Hastings Sampling

The preceding section showed that sampling a split according to Equations 17-20 results in an accurate MH framework. However, sampling the auxiliary variables from Equation 20 is not straightforward. This step is equivalent to sampling cluster parameters and labels for a 2-component

mixture model, which is known to be difficult. One typically samples from this space using an MCMC procedure. In fact, that is precisely what the restricted Gibbs sampler is doing. We therefore sample from Equation 20 by running a restricted Gibbs sampler for each newly proposed sub-cluster until they have burned-in. We monitor the data-likelihood for cluster $m$, $\overline{\mathcal{L}}_m = f_x(x_{\{m,\ell\}}; \overline{\theta}_{m,\ell}) \cdot f_x(x_{\{m,r\}}; \overline{\theta}_{m,r})$ and declare burn-in once $\overline{\mathcal{L}}_m$ begins to oscillate.

Furthermore, due to the implicit marginalization of auxiliary variables, the restricted Gibbs sampler and split moves that act on clusters that were not recently split do not depend on the proposed auxiliary variables. As such, these proposals can be computed before the auxiliary variables are even proposed. The sampling of auxiliary variables of a recently split cluster are *deferred* to the restricted Gibbs sampler while the other sampling steps are run concurrently. Once a set of proposed sub-clusters have burned-in, the corresponding clusters can be proposed to split again.

### 5.5 Merge Moves with Random Splits

The Hastings ratio for a merge depends on the proposed auxiliary variables for the reverse split. Since proposed splits are deterministic conditioned on the sub-cluster labels, the Hastings ratio will be zero if the proposed sub-cluster labels for a merge do not match those of the current clusters. We show in the supplement that as the number of data points grows, the acceptance ratio for a merge move quickly decays. With only 256 data points, the acceptance ratio for a merge proposal for 1000 trials in a 1D Gaussian mixture model did not exceed $10^{-16}$. We therefore approximate all merges with an automatic rejection. Unfortunately, this can lead to slow convergence in certain situations.

Fortunately, there is a very simple sampler that is good at proposing merges: a data-independent, random split proposal generated from the prior with a corresponding merge move. A split is constructed by sampling a random cluster, $c$, followed by a random partitioning of its data points form a Dirichlet-Multinomial. In general, these data-independent splits will be non-sensical and result in a rejection. However, merge moves are accepted with much higher probability than the sub-cluster merges. We refer the interested reader to the supplement for additional details.

## 6 Results

In this section, we compare the proposed method against other MCMC sampling algorithms. We consider three different versions of the proposed algorithm: using sub-clusters with and without super-clusters (SUBC and SUBC+SUPC) and an approximate method that does not wait for the convergence of sub-clusters to split (SUBC+SUPC APPROX). We note that while we do not expect this last version to converge to the correct distribution, empirical results show that it is similar in average performance. We compare the proposed methods against four other methods: the finite symmetric Dirichlet approximate model (FSD) with 100 components, a Rao-Blackwellized Gibbs sampler (GIBBS), a Rao-Blackwellized version of the original super-cluster work of [19] (GIBBS+SUPC), and the current state-of-the-art split/merge sampler [5] (GIBBS+SAMS). In our implementations, the concentration parameter is not resampled, though one could easily use a slice-sampler if desired.

We first compare these algorithms on synthetic Gaussian data with a Normal Inverse-Wishart prior. 100,000 data points are simulated from ten 2D Gaussian clusters. The average log likelihood for multiple sample paths obtained using the algorithms without parallelization for different numbers of initial clusters $K$ and concentration parameters $\alpha$ are shown in the first two columns of Figure 4. In this high data regime, $\alpha$ should have little effect on the resulting clusters. However, we find that the samplers without split/merge proposals (FSD, GIBBS, GIBBS+SC) perform very poorly when the initial number of clusters and the concentration parameter is small. We also find that the super-cluster method, GIBBS+SC, performs even worse than regular Gibbs sampling. This is likely due to super-clusters not being grouped by similar data, since data points not being able to move between different super-clusters can hinder convergence. In contrast, the proposed super-cluster method does not suffer from the same convergence problems, but is comparable to SUBC because there are a small number of clusters. Finally, the approximate sub-cluster method has significant gains when only one initial cluster is used, but performs approximately the same with more initial clusters.

Next we consider parallelizing the algorithms using 16 cores in the last column of Figure 4. The four inter-cluster parallelizable algorithms, SUBC, SUBC+SUPC, SUBC+SUPC APPROX, and FSD exhibit an order of magnitude speedup, while the the intra-cluster parallelizable algorithm

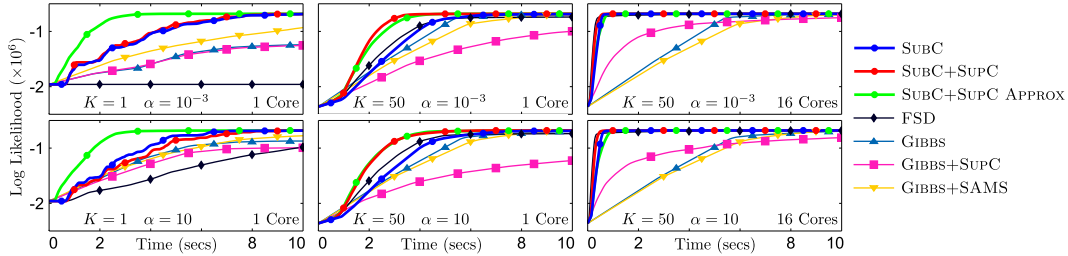

Figure 4: Synthetic data results for various initial clusters $K$, concentration parameters $\alpha$, and cores.

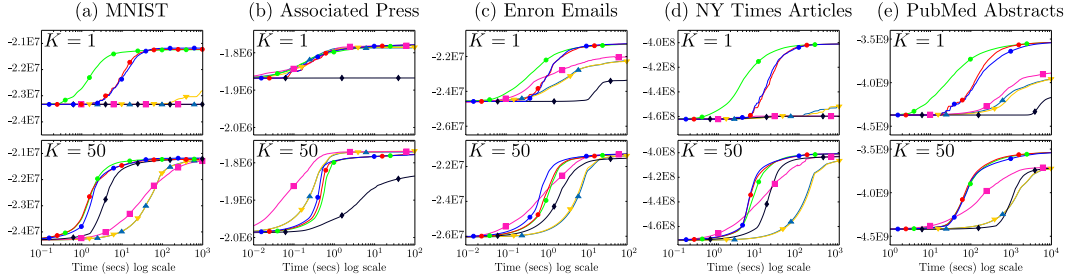

Figure 5: Log likelihood vs. computation time for real data. All parallel algorithms use 16 cores.

GIBBS+SUPC only has minor gains. As expected, parallelization does not aid the convergence of algorithms, only the speed at which they converge.

We now show results on real data. We test a Gaussian model with a Normal Inverse-Wishart prior on the MNIST dataset [16] by first running PCA on the 70,000 training and test images to 50 dimensions. Results on the MNIST dataset are shown in Figure 5a. We additionally test the algorithm on multinomial data with a Dirichlet prior on the following datasets: Associated Press [2] (2,246 documents and 10,473 dimension dictionary), Enron Emails [1] (39,861 documents and 28,102 dimension dictionary), New York Times articles [1] (300,000 documents and 102,660 dimension dictionary), and PubMed abstracts [1] (8,200,000 documents and 141,043 dimension dictionary). Results are shown in Figure 5b-e. In contrast to HDP models, each document is treated as a *single* draw from a multinomial distribution. We note that on the PubMed dataset, we had to increase the approximation of FSD to 500 components after observing that SUBC inferred approximately 400 clusters. On real data, it is clearly evident that the other algorithms have issues with convergence. In fact, in the allotted time, no algorithms besides the proposed methods converge to the same log likelihood with the two different initializations on the larger datasets. The presented sub-cluster methods converge faster to a better sample than other algorithms converge to a worse sample.

On the small, Associated Press dataset, the proposed methods actually perform slightly worse than the GIBBS methods. Approximately 20 clusters are inferred for this dataset, resulting in approximately 100 observations for each cluster. In these small data regimes, it is important to marginalize over as many variables as possible. We believe that because the GIBBS methods marginalize over the cluster parameters and weights, they achieve better performance as compared to the sub-cluster methods and FSD which explicitly instantiate them. This is not an issue with larger datasets.

## 7    Conclusion

We have presented a novel sampling algorithm for Dirichlet process mixture models. By alternating between a restricted Gibbs sampler and a split proposal, finite approximations to the DPMM are not needed and efficient inter-cluster parallelization can be achieved. Additionally, the proposed method for constructing splits based on fitting sub-clusters is, to our knowledge, the first parallelizable split algorithm for mixture models. Results on both synthetic and real data demonstrate that the speed of the sampler is orders of magnitude faster than other exact MCMC methods. Publicly available source code used in this work can be downloaded at http://people.csail.mit.edu/jchang7/.

# References

[1] K. Bache and M. Lichman. UCI machine learning repository, 2013.

[2] D. M. Blei, T. L. Griffiths, M. I. Jordan, and J. B. Tenenbaum. Hierarchical topic models and the nested Chinese restaurant process. In *NIPS*, 2003.

[3] D. M. Blei and M. I. Jordan. Variational inference for Dirichlet process mixtures. *Bayesian Analysis*, 1:121–144, 2005.

[4] C. A. Bush and S. N. MacEachern. A semiparametric Bayesian model for randomised block designs. *Biometrika*, 83:275–285, 1973.

[5] D. B. Dahl. An improved merge-split sampler for conjugate Dirichlet process mixture models. Technical report, University of Wisconsin - Madison Dept. of Statistics, 2003.

[6] M. D. Escobar and M. West. Bayesian density estimation and inference using mixtures. *Journal of the American Statistical Association*, 90(430):577–588, 1995.

[7] S. Favaro and Y. W. Teh. MCMC for normalized random measure mixture models. *Statistical Science*, 2013.

[8] T. S. Ferguson. A Bayesian analysis of some nonparametric problems. *The Annals of Statistics*, 1(2):209–230, 1973.

[9] P. J. Green and S. Richardson. Modelling heterogeneity with and without the Dirichlet process. *Scandinavian Journal of Statistics*, pages 355–375, 2001.

[10] W. K. Hastings. Monte Carlo sampling methods using Markov chains and their applications. *Biometrika*, 57(1):97–109, 1970.

[11] H. Ishwaran and L. F. James. Gibbs sampling methods for stick-breaking priors. *Journal of the American Statistical Association*, 96:161–173, 2001.

[12] H. Ishwaran and M. Zarepour. Exact and approximate sum-representations for the Dirichlet process. *Canadian Journal of Statistics*, 30:269–283, 2002.

[13] S. Jain and R. Neal. A split-merge Markov chain Monte Carlo procedure for the Dirichlet process mixture model. *Journal of Computational and Graphical Statistics*, 13:158–182, 2000.

[14] S. Jain and R. Neal. Splitting and merging components of a nonconjugate Dirichlet process mixture model. *Bayesian Analysis*, 2(3):445–472, 2007.

[15] K. Kurihara, M. Welling, and Y. W. Teh. Collapsed variational Dirichlet process mixture models. In *International Joint Conference on Artificial Intelligence*, 2007.

[16] Y. LeCun, L. Bottou, Y. Bengio, and P. Haffner. Gradient-based learning applied to document recognition. *Proceedings of the IEEE*, 86(11):2278–2324, 1998.

[17] P. Liang, M. I. Jordan, and B. Taskar. A permutation-augmented sampler for DP mixture models. In *Proceedings of the 24th international conference on Machine learning*, 2007.

[18] D. Lin, E. Grimson, and J. W. Fisher III. Construction of dependent Dirichlet processes based on Poisson processes. In *NIPS*, 2010.

[19] D. Lovell, R. P. Adams, and V. K. Mansingka. Parallel Markov chain Monte Carlo for Dirichlet process mixtures. In *Workshop on Big Learning, NIPS*, 2012.

[20] S. N. MacEachern. Estimating normal means with a conjugate style Dirichlet process prior. In *Communications in Statistics: Simulation and Computation*, 1994.

[21] S. N. MacEachern and P. Müller. Estimating mixture of Dirichlet process models. *Journal of Computational and Graphical Statistics*, 7(2):223–238, June 1998.

[22] R. Neal. Bayesian mixture modeling. In *Proceedings of the 11th International Workshop on Maximum Entropy and Bayesian Methods of Statistical Analysis*, 1992.

[23] R. Neal. Markov chain sampling methods for Dirichlet process mixture models. *Journal of Computational and Graphical Statistics*, 9(2):249–265, June 2000.

[24] O. Papaspiliopoulos and G. O. Roberts. Retrospective Markov chain Monte Carlo methods for Dirichlet process hierarchical models. *Biometrika*, 95(1):169–186, 2008.

[25] J. Pitman. Combinatorial stochastic processes. Technical report, U.C. Berkeley Dept. of Statistics, 2002.

[26] J. Sethuraman. A constructive definition of Dirichlet priors. *Statstica Sinica*, pages 639–650, 1994.

[27] E. B. Sudderth. *Graphical Models for Visual Object Recognition and Tracking*. PhD thesis, Massachusetts Institute of Technology, 2006.

[28] E. B. Sudderth, A. B. Torralba, W. T. Freeman, and A. S. Willsky. Describing visual scenes using transformed Dirichlet processes. In *NIPS*, 2006.

[29] Y. W. Teh, M. I. Jordan, M. J. Beal, and D. M. Blei. Hierarchical Dirichlet processes. *Journal of the American Statistical Association*, 101(476):1566–1581, 2006.

[30] M. West, P. Müller, and S. N. MacEachern. Hierarchical priors and mixture models, with application in regression and density estimation. *Aspects of Uncertainity*, pages 363–386, 1994.

[31] S. A. Williamson, A. Dubey, and E. P. Xing. Parallel Markov chain Monte Carlo for nonparametric mixture models. In *ICML*, 2013.

[32] E. P. Xing, R. Sharan, and M. I. Jordan. Bayesian haplotype inference via the Dirichlet process. In *ICML*, 2004.

